# Biasing Approximate Dynamic Programming with a Lower Discount Factor

**Marek Petrik**
Department of Computer Science
University of Massachusetts Amherst
Amherst, MA 01003
petrik@cs.umass.edu

**Bruno Scherrer**
LORIA Campus Scientifique B.P. 239
54506 Vandoeuvre-les-Nancy, France
bruno.scherrer@loria.fr

## Abstract

Most algorithms for solving Markov decision processes rely on a discount factor, which ensures their convergence. It is generally assumed that using an artificially low discount factor will improve the convergence rate, while sacrificing the solution quality. We however demonstrate that using an artificially low discount factor may significantly improve the solution quality, when used in approximate dynamic programming. We propose two explanations of this phenomenon. The first justification follows directly from the standard approximation error bounds: using a lower discount factor may decrease the approximation error bounds. However, we also show that these bounds are loose, thus their decrease does not entirely justify the improved solution quality. We thus propose another justification: when the rewards are received only sporadically (as in the case of Tetris), we can derive tighter bounds, which support a significant improvement in the solution quality with a decreased discount factor.

## 1 Introduction

Approximate dynamic programming methods often offer surprisingly good performance in practical problems modeled as Markov Decision Processes (MDP) [6, 2]. To achieve this performance, the parameters of the solution algorithms typically need to be carefully tuned. One such important parameter of MDPs is the discount factor $\gamma$. Discount factors are important in infinite-horizon MDPs, in which they determine how the reward is counted. The motivation for the discount factor originally comes from economic models, but has often no meaning in reinforcement learning problems. Nevertheless, it is commonly used to ensure that the rewards are bounded and that the Bellman operator is a contraction [8]. In this paper, we focus on the quality of the solutions obtained by approximate dynamic programming algorithms. For simplicity, we disregard the computational time, and use *performance* to refer to the quality of the solutions that are eventually obtained.

In addition to regularizing the rewards, using an artificially low discount factor sometimes has a significant effect on the performance of the approximate algorithms. Specifically, we have observed a significant improvement of approximate value iteration when applied to Tetris, a common reinforcement learning benchmark problem. The natural discount factor in Tetris is 1, since the received rewards have the same importance, independently of when received. Currently, the best results achieved with approximate dynamic programming algorithms are on average about 6000 lines removed in a single game [4, 3]. Our results, depicted in Figure 1, with approximate value iteration and standard features [1] show that setting the discount factor to $\gamma \in (0.84, 0.88)$ gives the best expected total number of removed lines, a bit more than 20000. That is five times the performance with discount factor of $\gamma = 1$ (about 4000). The improved performance for $\gamma \in (0.84, 0.88)$ is surprising, since computing a policy for this discount factor dramatically improves the return calculated with $\gamma = 1$.

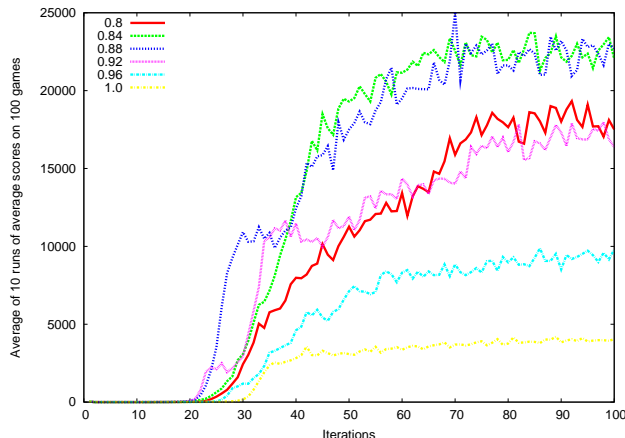

Figure 1: Performance of approximate value iteration on Tetris with different discount factors. For each value of $\gamma$, we ran the experiments 10 times and recorded the evolution of the score (the evaluation of the policy with $\gamma = 1$) on the 100 games, and averaged over 10 learning runs.

In this paper, we study why using a lower discount factor improves the quality of the solution with regard to a higher discount factor. First, in Section 2, we define the framework for our analysis. In Section 3 we analyze the influence of the discount factor on the standard approximation error bounds [2]. Then in Section 4 we argue that, in the context of this paper, the existing approximation error bounds are loose. Though these bounds may be tightened by a lower discount factor, they are not sufficient to explain the improved performance. Finally, to explain the improved performance, we identify a specific property of Tetris in Section 5 that enables the improvement. In particular, the rewards in Tetris are received sparsely, unlike the approximation error, which makes the value function less sensitive to the discount factor than the approximation error.

## 2 Framework and Notations

In this section we formalize the problem of adjusting the discount factor in approximate dynamic programming. We assume $\gamma$-discounted infinite horizon problems, with $\gamma < 1$. Tetris does not directly fit in this class, since its natural discount factor is $1$. It has been shown, however, that undiscounted infinite horizon problems with a finite total reward can be treated as discounted problems [7]. Blackwell optimality implies that there exists $\gamma^* < 1$ such that for all $\gamma > \gamma^*$ the $\gamma$-discounted problem and the undiscounted problem have the same optimal policy. We therefore treat Tetris as a discounted problem with a discount factor $\gamma^* < 1$ near one. The analysis is based on Markov decision processes, defined as follows.

**Definition 1.** A *Markov Decision Process* is a tuple $(S, A, P, r)$. $S$ is the set of states, $A$ is the set of actions, $P : S \times S \times A \mapsto [0, 1]$ is the transition function ($P(s', s, a)$ is the probability of transiting to state $s'$ from state $s$ given action $a$), and $r : S \times A \mapsto \mathbb{R}_+$ is a (non-negative) reward function.

We assume that the number of states and actions is finite, but possibly very large. For sake of simplicity, we also assume that the rewards are non-negative; our analysis can be extended to arbitrary rewards in a straight-forward way. We write $\|r\|_\infty$ to denote the maximal reward for any action and state.

Given a Markov decision process $(S, A, P, r)$ and some discount factor $\gamma$, the objective is to find a policy, i.e. a mapping $\pi : S \mapsto A$, with the *maximal* value from any initial states $s$. The value $v^\pi(s)$ of $\pi$ from state $s$ is defined as the $\gamma$-discounted infinite horizon return:

$$v^\pi(s) := \mathbf{E}\left[\sum_{t=0}^{\infty} \gamma^t r(s_t, a_t) \,\middle|\, s_0 = s, a_0 = \pi(s_0), \dots, a_t = \pi(s_t)\right].$$

It is well known [7, 2] that this problem can be solved by computing the optimal value function $v^*$, which is the fixed point of the Bellman operator $Lv = \max_\pi r_\pi + \gamma P_\pi v$. Here $r_\pi$ is the vector on $S$ with components $r(s, \pi(s))$ and $P^\pi$ is the stochastic matrix associated with a policy $\pi$.

We consider in this paper that the MDP is solved with 1) an approximate dynamic programming algorithm and 2) a different discount factor $\beta < \gamma$. In particular, our analysis applies to approximate value and policy iteration with existing error bounds. These methods invariably generate a sequence of approximate value functions, which we denote as $\tilde{v}_\beta$. Then, $\pi_\beta$ is a policy greedy with regard to the approximate value function $\tilde{v}_\beta$.

As we have two different discount factors, we use a subscript to denote the discount factor used in calculating the value. Let $\delta$ be a discount factor and $\pi$ any policy. We use $v_\delta^\pi$ to represent the value of policy $\pi$ calculated with the discount factor $\delta$; when $\pi$ is the optimal policy corresponding to the discount $\delta$, we will simply denote its value $v_\delta$. As mentioned above, our objective is to compare, for the discount factor $\gamma$, the value $v_\gamma$ of the optimal policy and the value $v_\gamma^{\pi_\beta}$. Here, $\pi_\beta$ is the policy derived from the approximate $\beta$-discount value. The following shows how this error may be decomposed in order to simplify the analysis. Most of our analysis is in terms of $L_\infty$ mainly because this is the most common measure used in the existing error bounds. Moreover, the results could be extended to $L_2$ norm in a rather straight-forward way without a qualitative difference in the results.

From the optimality of $v_\gamma$, $v_\gamma \geq v_\gamma^{\pi_\beta}$ and from the non-negativity of the rewards, it is easy to show that the value function is monotonous with respect to the discount factor, and therefore: $v_\gamma^{\pi_\beta} \geq v_\beta^{\pi_\beta}$. Thus $0 \leq v_\gamma - v_\gamma^{\pi_\beta} \leq v_\gamma - v_\beta^{\pi_\beta}$ and consequently:

$$e(\beta) := \|v_\gamma - v_\gamma^{\pi_\beta}\|_\infty \leq \|v_\gamma - v_\beta^{\pi_\beta}\|_\infty \leq \|v_\gamma - v_\beta\|_\infty + \|v_\beta - v_\beta^{\pi_\beta}\|_\infty = e_d(\beta) + e_a(\beta).$$

where $e_d(\beta) := \|v_\gamma - v_\beta\|_\infty$ denotes the *discount error*, and $e_a(\beta) := \|v_\beta - v_\beta^{\pi_\beta}\|_\infty$ the *approximation error*. In other words, a bound of the loss due to using $\pi_\beta$ instead of the optimal policy for discount factor $\gamma$ is the sum of the error on the optimal value function due to the change of discount and the error due to the approximation for discount $\beta$. In the remainder of the paper, we analyze each of these error terms.

## 3 Error Bounds

In this section, we develop a discount error bound and overview the existing approximation error bounds. We also show how these bounds motivate decreasing the discount factor in the majority of MDPs. First, we bound the discount error as follows.

**Theorem 2.** *The* discount error *due to using a discount factor $\beta$ instead of $\gamma$ is:*

$$e_d(\beta) = \|v_\gamma - v_\beta\|_\infty \leq \frac{\gamma - \beta}{(1 - \beta)(1 - \gamma)} \|r\|_\infty.$$

*Proof.* Let $L_\gamma$ and $L_\beta$ be the Bellman operators for the corresponding discount factors. We have now:

$$\begin{aligned}
\|v_\gamma - v_\beta\|_\infty &= \|L_\gamma v_\gamma - L_\beta v_\beta\|_\infty = \|L_\gamma v_\gamma - L_\beta v_\gamma + L_\beta v_\gamma - L_\beta v_\beta\|_\infty \\
&\leq \|L_\gamma v_\gamma - L_\beta v_\gamma\|_\infty + \|L_\beta v_\gamma - L_\beta v_\beta\|_\infty \leq \|L_\gamma v_\gamma - L_\beta v_\gamma\|_\infty + \beta\|v_\gamma - v_\beta\|_\infty
\end{aligned}$$

Let $P_\gamma, r_\gamma$ and $P_\beta, r_\beta$ be the transition matrices and rewards of policies greedy with regard to $v_\gamma$ for $\gamma$ and $\beta$ respectively. Then we have:

$$\begin{aligned}
L_\gamma v_\gamma - L_\beta v_\gamma &= (\gamma P_\gamma v_\gamma + r_\gamma) - (\beta P_\beta v_\gamma + r_\beta) \leq (\gamma - \beta) P_\gamma v_\gamma \\
L_\gamma v_\gamma - L_\beta v_\gamma &= (\gamma P_\gamma v_\gamma + r_\gamma) - (\beta P_\beta v_\gamma + r_\beta) \geq (\gamma - \beta) P_\beta v_\gamma.
\end{aligned}$$

Finally, the bound follows from above as:

$$\|v_\gamma - v_\beta\|_\infty \leq \frac{1}{1 - \beta} \max\{\|(\gamma - \beta) P_\gamma v_\gamma\|_\infty, \|(\gamma - \beta) P_\beta v_\gamma\|_\infty\} \leq \frac{\gamma - \beta}{(1 - \gamma)(1 - \beta)} \|r\|_\infty.$$

$\square$

*Remark* 3. This bound is trivially tight, that is there exists a problem for which the bound reduces to equality. It is however also straightforward to construct a problem in which the bound is not tight.

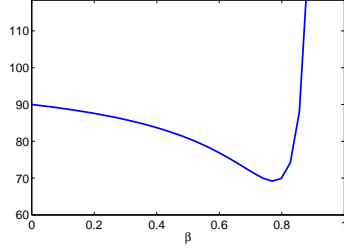

Figure 2: Example $e(\beta)$ function in a problem with $\gamma = 0.9$ and $\epsilon = 0.01$ and $\|r\|_\infty = 10$.

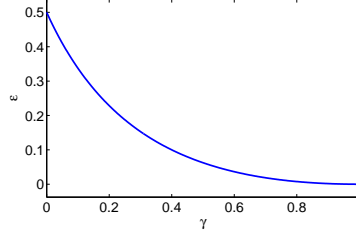

Figure 3: The dependence of $\epsilon$ on $\gamma$ needed for the improvement in Proposition 6.

## 3.1 Approximation Error Bound

We now discuss the dependence of the approximation error $e_a(\beta)$ on the discount factor $\beta$. Approximate dynamic programming algorithms like approximate value and policy iteration build a sequence of value functions $(\tilde{v}_\beta^k)_{k\geq 0}$ with $\pi_\beta^k$ being the policy greedy with respect to $\tilde{v}_\beta^k$. These algorithms are approximate because at each iteration the value $\tilde{v}_\beta^k$ is an approximation of some target value $v_\beta^k$, which is hard to compute. The analysis of [2] (see Section 6.5.3 and Proposition 6.1 for value iteration, and Proposition 6.2 for policy iteration) bounds the loss of using the policies $\pi_\beta^k$ instead of the optimal policy:

$$\limsup_{k\to\infty} \|v_\beta - v_\beta^{\pi_\beta^k}\|_\infty \leq \frac{2\beta}{(1-\beta)^2} \sup_k \|\tilde{v}_\beta^k - v_\beta^k\|_\infty. \tag{1}$$

To completely describe how Eq. (1) depends on the discount factor, we need to bound the one-step approximation error $\|\tilde{v}_\beta^k - v_\beta^k\|$ in terms of $\beta$. Though this specific error depends on the particular approximation framework used and is in general difficult to estimate, we propose to make the following assumption.

**Assumption 4.** There exists $\epsilon \in (0, 1/2)$, such that for all $k$, the *single-step* approximation error is bounded by:

$$\|\tilde{v}_\beta^k - v_\beta^k\|_\infty \leq \frac{\epsilon}{1-\beta}\|r\|_\infty.$$

We consider only $\epsilon \leq 1/2$ because the above assumption holds with $\epsilon = 1/2$ and the trivial constant approximation $\tilde{v}_\beta^k = \|r\|_\infty/2$.

*Remark* 5. Alternatively to Assumption 4, we could assume that the approximation error is constant in the discount factor $\beta$, i.e. $\|\tilde{v}_\beta^k - v_\beta^k\|_\infty \leq \epsilon = O(1)$ for some $\epsilon$ for all $\beta$. We believe that such a bound is unlikely in practice. To show that, consider an MDP with two states $s_0$ and $s_1$, and a single action. The transitions loop from each state to itself, and the rewards are $r(s_0) = 0$ and $r(s_1) = 1$. Assume a linear least-squares approximation with basis $M = [1/\sqrt{2}; 1/\sqrt{2}]$. The approximation error in terms of $\beta$ is: $1/2(1-\beta) = O(1/(1-\beta))$.

If Assumption 4 holds, we see from Eq. (1) that the approximation error $e_a$ is bounded as:

$$e_a(\beta) \leq \frac{2\beta}{(1-\beta)^3}\epsilon\|r\|_\infty.$$

## 3.2 Global Error Bound

Using the results above, and considering that Assumption 4 holds, the cumulative error bound when using approximate dynamic programming with a discount factor $\beta < \gamma$ is:

$$e(\beta) = e_a(\beta) + e_d(\beta) \leq \frac{\gamma - \beta}{(1-\beta)(1-\gamma)}\|r\|_\infty + \frac{2\beta}{(1-\beta)^3}\epsilon\|r\|_\infty.$$

An example of this error bound is shown in Figure 2: the bound is minimized for $\beta \simeq 0.8 < \gamma$. This is because the approximation error decreases rapidly in comparison with the increasing discount error. More generally, the following proposition suggests how we should choose $\beta$.

**Proposition 6.** *If the approximation factor $\epsilon$ introduced in Assumption 4 is sufficiently large, precisely if $\epsilon > (1-\gamma)^2/2(1+2\gamma)$, then the best error bound $e(\beta)$ will be achieved for the discount factor $\beta = (2\epsilon+1) - \sqrt{(2\epsilon+1)^2 + (2\epsilon-1)} < \gamma$.*

Figure 3 shows the approximation error fraction necessary to improve the performance. Notice that the fraction decreases rapidly when $\gamma \to 1$.

*Proof.* The minimum of $\beta \mapsto e(\beta)$ can be derived analytically by taking its derivative:

$$
\begin{aligned}
e'(\beta) &= -(1-\beta)^{-2}\|r\|_\infty + (1-\beta)^{-3}2\|r\|_\infty\epsilon + (-3)2\beta(-1)(1-\beta)^{-4}\|r\|_\infty\epsilon \\
&= \frac{(1-\beta)^2 + 2(1-\beta)\epsilon + 6\beta\epsilon}{(1-\beta)^4}\|r\|_\infty = \frac{-\beta^2 + 2(2\epsilon+1)\beta + 2\epsilon - 1}{(1-\beta)^4}\|r\|_\infty.
\end{aligned}
$$

So we want to know when $\beta \mapsto -1/2\beta^2 + (2\epsilon+1)\beta + 1/2(2\epsilon-1)$ equals 0. The discriminant $\Delta = (2\epsilon+1)^2 + (2\epsilon-1) = 2\epsilon(2\epsilon+3)$ is always positive. Therefore $e'(\beta)$ equals 0 for the points $\beta_- = (2\epsilon+1) - \sqrt{\Delta}$ and $\beta_+ = (2\epsilon+1) + \sqrt{\Delta}$ and is positive in between and negative outside. This means that $\beta_-$ is a local minimum of $e$ and $\beta_+$ a local maximum.
It is clear that $\beta_+ > 1 > \gamma$. From the definition of $\Delta$ and the fact (cf Assumption 4) that $\epsilon \le 1/2$, we see that $\beta_- \ge 0$. Then, the condition $\beta_- < \gamma$ is satisfied if and only if:

$$
\begin{aligned}
\beta_- < \gamma &\Leftrightarrow (2\epsilon+1) - \sqrt{(2\epsilon+1)^2 + (2\epsilon-1)} < \gamma \Leftrightarrow 1 - \sqrt{1 + \frac{2\epsilon-1}{(2\epsilon+1)^2}} < \frac{\gamma}{2\epsilon+1} \\
&\Leftrightarrow 1 - \frac{\gamma}{2\epsilon+1} < \sqrt{1 + \frac{2\epsilon-1}{(2\epsilon+1)^2}} \Leftrightarrow 1 - 2\frac{\gamma}{2\epsilon+1} + \frac{\gamma^2}{(2\epsilon+1)^2} < 1 + \frac{2\epsilon-1}{(2\epsilon+1)^2} \\
&\Leftrightarrow -2\gamma(2\epsilon+1) + \gamma^2 < 2\epsilon - 1 \Leftrightarrow \frac{(1-\gamma)^2}{1+2\gamma} < 2\epsilon
\end{aligned}
$$

where the inequality holds after squaring, since both sides are positive. $\quad\square$

## 4 Bound Tightness

We show in this section that the bounds on the approximation error $e_a(\beta)$ are very loose for $\beta \to 1$ and thus the analysis above does not fully explain the improved performance. In particular, there exists a naive bound on the approximation error that is dramatically tighter than the standard bounds when $\beta$ is close to 1.

**Lemma 7.** *There exists a constant $c \in \mathbb{R}_+$ such that for all $\beta$ we have $\|v_\beta - \tilde{v}_\beta\|_\infty \le c/(1-\beta)$.*

*Proof.* Let $P^*, r^*$ and $\hat{P}, \hat{r}$ be the transition reward functions of the optimal approximate policies respectively. The functions may depend on the discount factor, but we omit that to simplify the notation. Then the approximation error is:

$$
\|v_\beta - \hat{v}_\beta\|_\infty = \|(I - \beta P^*)^{-1}r^* - (I - \beta\hat{P})^{-1}\hat{r}\|_\infty \le \frac{1}{1-\beta}\left(\|r^*\|_\infty + \|\hat{r}\|_\infty\right).
$$

Thus setting $c = 2\max_\pi \|r_\pi\|_\infty$ proves the lemma. $\quad\square$

Lemma 7 implies that for every MDP, there exists a discount factor $\beta$, such that Eq. (1) is not tight. Consider even that the *single-step* approximation error is bounded by a constant, such that $\limsup_{k\to\infty}\|\tilde{v}_\beta^k - v_\beta^k\|_\infty \le \epsilon$. This is impractical, as discussed in Remark 5, but it tightens the bound. Such a bound implies that: $e_a(\beta) \le 2\beta\epsilon/(1-\beta)^2$. From Lemma 7, this bound is loose when $\frac{2\beta}{(1-\beta)^2}\epsilon > \frac{c}{1-\beta}$. Thus we have that there exists $\beta < 1$ for which the standard approximation error bounds are loose, whenever $\epsilon > 0$. The looseness of the bound will be more apparent in problems with high discount factors. For example in the MDP formulation of Blackjack [5] the discount factor $\gamma = 0.999$, in which case the error bound may overestimate the true error by a factor up to $1/(1-\gamma) = 1000$.

The looseness of the approximation error bounds may seem to contradict Example 6.4 in [2], which shows that Eq. (1) is tight. The discrepancy is because in our analysis we assume that the MDP has

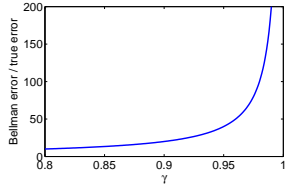

Figure 4: Looseness of the Bellman error bound.

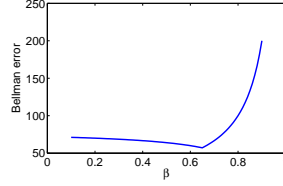

Figure 5: Bellman error bound as a function of $\beta$ for a problem with $\gamma = 0.9$.

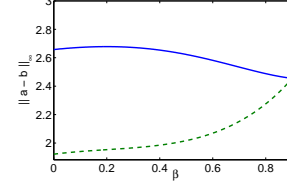

Figure 6: The approximation error with $a = \tilde{v}_\beta$ and $b = v_\gamma$.

fixed rewards and number of states, while the example in [2] assumes that the reward depends on the discount factor and the number of states is potentially infinite. Another way to put it is to say that Example 6.4 shows that for any discount factor $\beta$ there exists an MDP (which depends on $\beta$) for which the bound Eq. (1) is tight. We, on the other hand, show that there does not exist a fixed MDP such that for all discount factor $\beta$ the bound Eq. (1) is tight.

Proposition 6 justifies the improved performance with a lower discount factor by a more rapid decrease in $e_a$ with $\beta$ than the increase in $e_d$. The naive bound from Lemma 7 however shows that $e_a$ may scale with $1/(1 - \beta)$, the same as $e_d$. As a result, while the approximation error will decrease, it may not be sufficient to offset the increase in the discount error.

Some of the standard approximation error bound may be tightened by using a lower discount factor. For example consider the standard a-posteriori approximation error bound for the value function $\tilde{v}_\beta$ [7] :

$$\|v_\beta - v_\beta^{\tilde{\pi}}\|_\infty \leq \frac{1}{1 - \beta}\|L_\beta \tilde{v}_\beta - \tilde{v}_\beta\|_\infty,$$

where $\tilde{\pi}_\beta$ is greedy with respect to $\tilde{v}_\beta$. This bound is widely used and known as *Bellman error* bound. The following example demonstrates that the Bellman error bound may also be loose for $\beta$ close to 1:

$$P_1 = \begin{pmatrix} 1 & 0 \\ 0 & 1 \end{pmatrix} \quad P_2 = \begin{pmatrix} 0 & 1 \\ 0 & 1 \end{pmatrix} \quad r_1 = \begin{pmatrix} 1 & 2 \end{pmatrix} \quad r_2 = \begin{pmatrix} 2 & 2 \end{pmatrix}$$

Assume that the current value function is the value of a policy with the transition matrix and reward $P_1, r_1$, while the optimal policy has the transition matrix and reward $P_2, r_2$. The looseness of the bound is depicted in Figure 4. The approximation error bound scales with $\frac{1}{(1-\gamma)^2}$, while the true error scales with $\frac{1}{1-\gamma}$. As a result, for $\gamma = 0.999$, the bound is 1000 times the true error value in this example. The intuitive reason for the looseness of the bound is that the bound treats each state as recurrent, even when is it transient.

The global error bound may be also tightened by using a lower discount factor $\beta$ as follows:

$$\|v_\gamma - v_\gamma^{\tilde{\pi}_\beta}\|_\infty \leq \frac{1}{1 - \beta}\|L_\beta \tilde{v}_\beta - \tilde{v}_\beta\|_\infty + \frac{\gamma - \beta}{(1 - \beta)(1 - \gamma)}\|r\|_\infty.$$

Finding the discount factor $\beta$ that minimizes this error is difficult, because the function may not be convex or differentiable. Thus the most practical method is a sub-gradient optimization method. The global error bound the MDP example above is depicted in Figure 5.

## 5 Sparse Rewards

In this section, we propose an alternative explanation for the performance improvement in Tetris that does not rely on the loose approximation error bounds. A specific property of Tetris is that the rewards are not received in every step, i.e. they are sparse. The value function, on the other hand, is approximated in every step. As a result, the return should be less sensitive to the discount factor than the approximation error. Decreasing the discount factor will thus reduce the approximation error more significantly than it increases the discount error. The following assumption formalizes this intuition.

**Assumption 8** (Sparse rewards). There exists an integer $q$ such that for all $m \geq 0$ and all instantiations $r_i$ with non-zero probability: $\sum_{i=0}^{m} r_i \leq \lfloor m/q \rfloor$ and $r_i \in \{0, 1\}$.

Now define $u_\beta = \sum_{i=0}^\infty \beta^i t_i$, where $t_i = 1$ when $i \equiv 0 \mod q$. Then let $I_m = \{i \,|\, r_i = 1, i \le m\}$ and $J_m = \{j \,|\, t_j = 1, j \le m\}$ and let $I = I_\infty$ and $J = J_\infty$. From the definition, these two sets satisfy that $|I_m| \le |J_m|$. First we show the following lemma.

**Lemma 9.** *Given sets $I_m$ and $J_m$, there exists an injective function $f : I \to J$, such that $f(i) \le i$.*

*Proof.* By induction on $m$. The base case $m = 0$ is trivial. For the inductive case, consider the following two cases: 1) $r_{m+1} = 0$. From the inductive assumption, there exists a function that maps $I_m$ to $J_m$. Now, this is also an injective function that maps $I_{m+1} = I_m$ to $J_{m+1}$. 2) $r_{m+1} = 1$. Let $j^* = \max J_{m+1}$. Then if $j^* = m + 1$ then the function $f : I_m \to J_m$ can be extended by setting $f(m+1) = j^*$. If $j^* \le m$ then since $|J_{m+1}| - 1 = |J_{j^*-1}| \ge |I_m|$, such an injective function exists from the inductive assumption. □

In the following, let $R_i$ be the random variable representing the reward received in step $i$. It is possible to prove that the discount error scales with a coefficient that is lower than in Theorem 2:

**Theorem 10.** *Let $\beta \le \gamma - \phi$, let $k = -\log(1 - \gamma)/(\log(\gamma) - \log(\gamma - \phi))$, and let $\rho = \mathbf{E}\left[\sum_{i=0}^k \gamma^i R_i\right]$. Then assuming the reward structure as defined in Assumption 8 we have that:*

$$\|v_\gamma - v_\beta\|_\infty \le \gamma^k \|u_\gamma - u_\beta\|_\infty + \rho \le \frac{\gamma^k(\gamma^q - \beta^q)}{(1 - \gamma^q)(1 - \beta^q)} + \rho.$$

*Proof.* Consider $\pi$ be the optimal policy for the discount factor $\gamma$. Then we have: $0 \le v_\gamma - v_\beta \le v_\gamma^\pi - v_\beta^\pi$. In the remainder of the proof, we drop the superscript $\pi$ for simplicity, that is $v_\beta = v_\beta^\pi$, not the optimal value function. Intuitively, the proof is based on "moving" the rewards to earlier steps to obtain a regular rewards structure. A small technical problem with this approach is that moving the rewards that are close to the initial time step decreases the bound. Therefore, we treat these rewards separately within the constant $\rho$. First, we show that for $f(i) \ge k$, we have that $\gamma^i - \beta^i \le \gamma^{f(i)} - \beta^{f(i)}$. Let $j = f(i) = i - k$, for some $k \ge 0$. Then:

$$
\begin{aligned}
\gamma^j - \beta^j &\ge \gamma^{j+k} - \beta^{j+k} \\
j &\ge \max_{\beta \in [0, \gamma - \phi]} \frac{\log(1 - \beta^k) - \log(1 - \gamma^k)}{\log(\gamma) - \log(\beta)} \ge \frac{-\log(1 - \gamma^k)}{\log(\gamma) - \log(\gamma - \phi)},
\end{aligned}
$$

with the maximization used to get a sufficient condition independent of $\beta$. Since the function $f$ maps only at most $\lfloor k/q \rfloor$ values of $I_m$ to $j < k$, there is such $|I_z| = k$, that $\forall x \in I_m \setminus I_z \; f(x) \ge k$. Then we have for $j > k$:

$$
\begin{aligned}
0 \le v_\gamma - v_\beta &= \lim_{m \to \infty} \mathbf{E}\left[\sum_{i \in I_m \setminus I_z}(\gamma^i - \beta^i)\right] \le \rho + \lim_{m \to \infty} \mathbf{E}\left[\sum_{i=1\ldots m \wedge f(i) \ge k}(\gamma^{f(i)} - \beta^{f(i)})t_{f(i)}\right] \\
&\le \rho + \sum_{j=k}^\infty (\gamma^j - \beta^j)t_j = \rho + \gamma^k(u_\gamma - u_\beta).
\end{aligned}
$$

□

Because the playing board in Tetris is 10 squares wide, and each piece has 4 squares, it takes on average 2.5 moves to remove a line. Since Theorem 10 applies only to integer values of $q$, we use a Tetris formulation in which dropping each piece requires two steps. A proper Tetris action is taken in the first step, and there is no action in the second one. To make this model identical to the original formulation, we change the discount factor to $\gamma^{\frac{1}{2}}$. Then the upper bound from Theorem 10 on the discount error is: $\|v_\gamma - v_\beta\|_\infty \le \gamma^k(\gamma^{2.5} - \beta^{2.5})/(1 - \gamma^{2.5})(1 - \beta^{2.5}) + \rho$, Notice that $\rho$ is a constant; it is independent of the new discount factor $\beta$.

The sparse rewards property can now be used to motivate the performance increase, even if the approximation error is bounded by $\epsilon/(1 - \beta)$ instead of by $\epsilon/(1 - \beta)^3$ (as Lemma 7 suggests). The approximation error bound will not, in most cases, satisfy the sparsity assumption, as the errors are typically distributed almost uniformly over the state space and is received in every step as a result. Therefore, for sparse rewards, the discount error increase will typically be offset by the larger decrease in the approximation error.

The cumulative error bounds derived above predict it is beneficial to reduce the discount factor to $\beta$ when:

$$\|v_\gamma - v_\beta\|_\infty \leq \gamma^k \frac{(\gamma^{2.5} - \beta^{2.5})}{(1 - \gamma^{2.5})(1 - \beta^{2.5})} + \rho + \frac{\epsilon}{1 - \beta} < \frac{\epsilon}{1 - \gamma}.$$

The effective discount factor $\gamma^*$ in Tetris is not known, but consider for example that it is $\gamma^* = 0.99$. Assuming $\phi = 0.1$ we have that $k = 48$, which means that the first $\lfloor 48/2.5 \rfloor$ rewards must be excluded, and included in $\rho$. The bounds then predict that for $\epsilon \geq 0.4$ the performance of approximate value iteration may be expected to improve using $\beta \leq \gamma - \phi$.

We end by empirically illustrating the influence of reward sparsity in a general context. Consider a simple 1-policy, 7-state chain problem. Consider two reward instances, one with a single reward of 1, and the other with randomly generated rewards. We show the comparison of the effects of a lower discount factor of these two examples in Figure 6. The dotted line represents the global error with sparse rewards, and the solid line represents the cumulative error with dense rewards. Sparsity of rewards makes a decrease of the discount factor more interesting.

## 6  Conclusion and Future Work

We show in this paper that some common approximation error bounds may be tightened with a lower discount factor. We also identified a class of problems in which a lower discount factor is likely to increase the performance of approximate dynamic programming algorithms. In particular, these are problems in which the rewards are received relatively sparsely. We concentrated on a theoretical analysis of the influence of the discount factor, not on the specific methods which could be used to determine a discount factor. The actual dependence of the performance on the discount factor may be non-trivial, and therefore hard to predict based on simple bounds. Therefore, the most practical approach is to first predict an improving discount factor based on the theoretical predictions, and then use line search to find a discount factor that ensures good performance. This is possible since the discount factor is a single-dimensional variable with a limited range.

The central point of our analysis is based on bounds that are in general quite loose. An important future direction is to analyze the approximation error more carefully. We shall do experiments in order to see if we can have some insight on the form (i.e. the distribution) of the error for several settings (problems, approximation architecture). If such errors follow some law, it might be interesting to see whether it helps to tighten the bounds.

**Acknowledgements**   This work was supported in part by the Air Force Office of Scientific Research Grant No. FA9550-08-1-0171 and by the National Science Foundation Grant No. 0535061. The first author was also supported by a University of Massachusetts Graduate Fellowship.

## References

[1] Dimitri P. Bertsekas and Sergey Ioffe. Temporal differences-based policy iteration and applications in neuro-dynamic programming. Technical Report LIDS-P-2349, LIDS, 1997.

[2] Dimitri P. Bertsekas and John N. Tsitsiklis. *Neuro-dynamic programming*. Athena Scientific, 1996.

[3] V.F. Farias and B. Van Roy. *Probabilistic and Randomized Methods for Design Under Uncertainty*, chapter 6: Tetris: A Study of Randomized Constraint Sampling. Springer-Verlag, 2006.

[4] Sham Machandranath Kakade. A Natural Policy Gradient. In *Advances in neural information processing systems*, pages 1531–1538. MIT Press, 2001.

[5] Ronald Parr, Lihong Li, Gavin Taylor, Christopher Painter-Wakefield, and Michael L. Littman. An analysis of linear models, linear value function approximation, and feature selection for reinforcement learning. In *International Conference on Machine Learning*, 2008.

[6] Warren B. Powell. *Approximate Dynamic Programming*. Wiley-Interscience, 2007.

[7] Martin L. Puterman. *Markov decision processes: Discrete stochastic dynamic programming*. John Wiley & Sons, Inc., 2005.

[8] Richard S. Sutton and Andrew Barto. *Reinforcement learning*. MIT Press, 1998.